# Unsupervised Learning by Convex and Conic Coding

**D. D. Lee and H. S. Seung**
Bell Laboratories, Lucent Technologies
Murray Hill, NJ 07974
{ddlee|seung}@bell-labs.com

## Abstract

Unsupervised learning algorithms based on convex and conic encoders are proposed. The encoders find the closest convex or conic combination of basis vectors to the input. The learning algorithms produce basis vectors that minimize the reconstruction error of the encoders. The convex algorithm develops locally linear models of the input, while the conic algorithm discovers features. Both algorithms are used to model handwritten digits and compared with vector quantization and principal component analysis. The neural network implementations involve feedback connections that project a reconstruction back to the input layer.

## 1  Introduction

Vector quantization (VQ) and principal component analysis (PCA) are two widely used unsupervised learning algorithms, based on two fundamentally different ways of encoding data. In VQ, the input is encoded as the index of the closest prototype stored in memory. In PCA, the input is encoded as the coefficients of a linear superposition of a set of basis vectors. VQ can capture nonlinear structure in input data, but is weak because of its highly localized or "grandmother neuron" representation. Many prototypes are typically required to adequately represent the input data when the number of dimensions is large. On the other hand, PCA uses a distributed representation so it needs only a small number of basis vectors to model the input. Unfortunately, it can only model linear structures.

Learning algorithms based on convex and conic encoders are introduced here. These encoders are less constrained than VQ but more constrained than PCA. As a result,

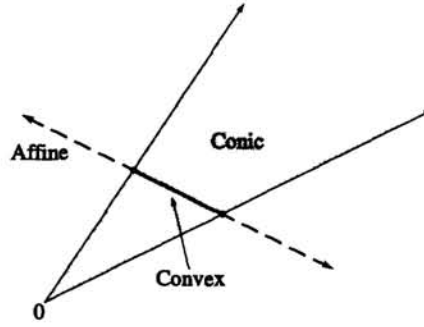

Figure 1: The affine, convex, and conic hulls for two basis vectors.

they are able to produce sparse distributed representations that are efficient to compute. The resulting learning algorithms can be understood as approximate matrix factorizations and can also be implemented as neural networks with feedforward and feedback connections between neurons.

## 2   Affine, convex, conic, and point encoding

Given a set of basis vectors $\{\vec{w}_a\}$, the linear combination $\sum_{a=1}^{r} v_a \vec{w}_a$ is called

$$
\left.\begin{array}{l}
\text{affine} \\
\text{convex} \\
\text{conic}
\end{array}\right\}
\quad \text{if} \quad
\left\{\begin{array}{ll}
 & \sum_a v_a = 1\,, \\
v_a \geq 0\,, & \sum_a v_a = 1\,, \\
v_a \geq 0\,.
\end{array}\right.
$$

The complete set of affine, convex, and conic combinations are called respectively the affine, convex, and conic hulls of the basis. These hulls are geometrically depicted in Figure 1. The convex hull contains only interpolations of the basis vectors, whereas the affine hull contains not only the convex hull but also linear extrapolations. The conic hull also contains the convex hull but is not constrained to stay within the set $\sum_a v_a = 1$. It extends to any nonnegative combination of the basis vectors and forms a cone in the vector space.

Four encoders are considered in this paper. The convex and conic encoders are novel, and find the nearest point to the input $\vec{x}$ in the convex and conic hull of the basis vectors. These encoders are compared with the well-known affine and point encoders. The affine encoder finds the nearest point to $\vec{x}$ in the affine hull and is equivalent to the encoding in PCA, while the point encoder or VQ finds the nearest basis vector to the input. All of these encoders minimize the *reconstruction error*:

$$
\min_{v_a} \left\| \vec{x} - \sum_{a=1}^{r} v_a \vec{w}_a \right\|^2. \tag{1}
$$

The constraints on $v_a$ for the convex, conic, and affine encoders were described above. Point encoding can be thought of as a heavily constrained optimization of Eq. (1): a single $v_a$ must equal unity while all the rest vanish.

Efficient algorithms exist for computing all of these encodings. The affine and point encoders are the fastest. Affine encoding is simply a linear transformation of the input vector. Point encoding is a nonlinear operation, but is computationally simple

since it involves only a minimum distance computation. The convex and conic encoders require solving a quadratic programming problem. These encodings are more computationally demanding than the affine and point encodings; nevertheless, polynomial time algorithms do exist. The tractability of these problems is related to the fact that the cost function in Eq. (1) has no local minima on the convex domains in question. These encodings should be contrasted with computationally inefficient ones. A natural modification of the point encoder with combinatorial expressiveness can be obtained by allowing $\vec{v}$ to be any vector of zeros and ones [1, 2]. Unfortunately, with this constraint the optimization of Eq. (1) becomes an integer programming problem and is quite inefficient to solve.

The convex and conic encodings of an input generally contain coefficients $v_a$ that vanish, due to the nonnegativity constraints in the optimization of Eq. (1). This method of obtaining sparse encodings is distinct from the method of simply truncating a linear combination by discarding small coefficients [3].

## 3 Learning

There correspond learning algorithms for each of the encoders described above that minimize the average reconstruction error over an ensemble of inputs. If a training set of $m$ examples is arranged as the columns of a $N \times m$ matrix $X$, then the learning and encoding minimization can be expressed as:

$$\min_{W,V} \|X - WV\|^2 \tag{2}$$

where $\|X\|^2$ is the summed squares of the elements of $X$. Learning and encoding can thus be described as the approximate factorization of the data matrix $X$ into a $N \times r$ matrix $W$ of $r$ basis vectors and a $r \times m$ matrix $V$ of $m$ code vectors.

Assuming that the input vectors in $X$ have been scaled to the range $[0, 1]$, the constraints on the optimizations in Eq. (2) are given by:

**Affine:**   $0 \leq W_{ia} \leq 1,$                  $\sum_a V_{a\mu} = 1$
**Convex:** $0 \leq W_{ia} \leq 1,$   $V_{a\mu} \geq 0,$   $\sum_a V_{a\mu} = 1$
**Conic:**    $0 \leq W_{ia} \leq 1,$   $V_{a\mu} \geq 0.$

The nonnegativity constraints on $W$ and $V$ prevent cancellations from occurring in the linear combinations, and their importance will be seen shortly. The upper bound on $W$ is chosen such that the basis vectors are normalized in the same range as the inputs $X$. We noted earlier that the computation for encoding is tractable since the cost function Eq. (2) is a quadratic function of $V$. However, when considered as a function of both $W$ and $V$, the cost function is quartic and finding its global minimum for learning can be very difficult. The issue of local minima is discussed in the following example.

## 4 Example: modeling handwritten digits

We applied **Affine, Convex, Conic,** and VQ learning to the USPS database [4], which consists of examples of handwritten digits segmented from actual zip codes. Each of the 7291 training and 2007 test images were normalized to a $16 \times 16$ grid

VQ                                                        Affine (PCA)

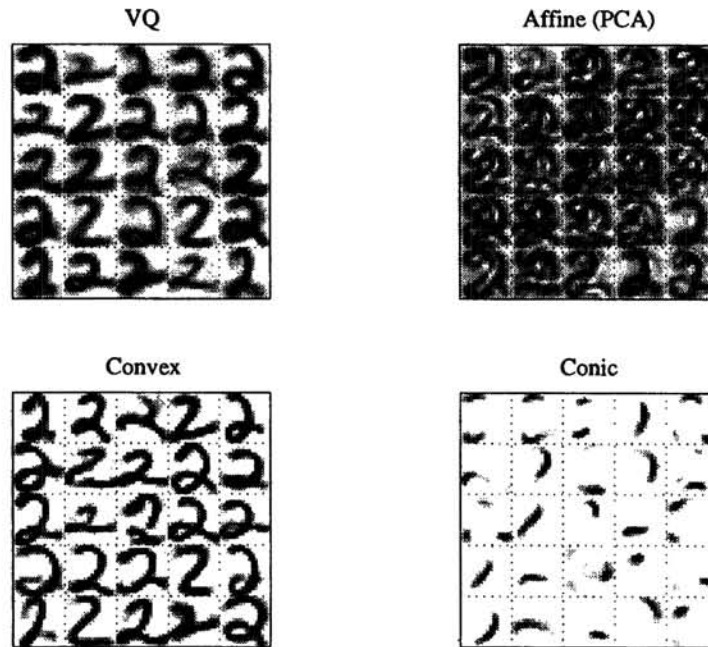

Convex                                                    Conic

Figure 2: Basis vectors for "2" found by VQ, **Affine**, **Convex**, and **Conic** learning.

with pixel intensities in the range $[0, 1]$. There were noticeable segmentation errors resulting in unrecognizable digits, but these images were left in both the training and test sets. The training examples were segregated by digit class and separate basis vectors were trained for each of the classes using the four encodings. Figure 2 shows our results for the digit class "2" with $r = 25$ basis vectors.

The $k$-means algorithm was used to find the VQ basis vectors shown in Figure 2. Because the encoding is over a discontinuous and highly constrained space, there exist many local minima to Eq. (2). In order to deal with this problem, the algorithm was restarted with various initial conditions and the best solution was chosen. The resulting basis vectors look like "2" templates and are blurry because each basis vector is the mean of a large number of input images.

**Affine** determines the affine space that best models the input data. As can be seen in the figure, the individual basis vectors have no obvious interpretation. Although the space found by **Affine** is unique, its representation by basis vectors is degenerate. Any set of $r$ linearly independent vectors drawn from the affine space can be used to represent it. This is due to the fact that the product $WV$ is invariant under the transformation $W \to WS$ and $V \to S^{-1}V$.[1]

**Convex** finds the $r$ basis vectors whose convex hull best fits the input data. The optimization was performed by alternating between projected gradient steps of $W$ and $V$. The constraint that the column sums of $V$ equal unity was implemented

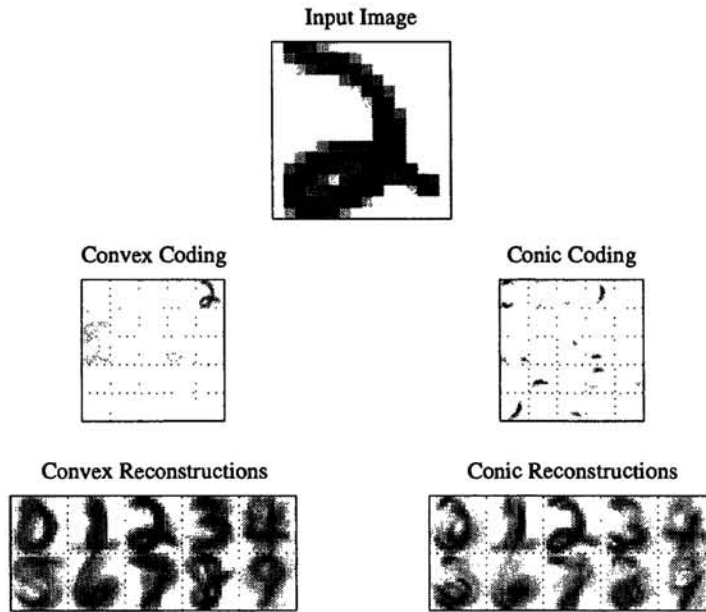

Figure 3: Activities and reconstructions of a "2" using conic and convex coding.

by adding a quadratic penalty term. In contrast to **Affine**, the basis vectors are interpretable as templates and are less blurred than those found by VQ. Many of the elements of $W$ and also of $V$ are zero at the minimum. This eliminates many invariant transformations $S$, because they would violate the nonnegativity constraints on $W$ and $V$. From our simulations, it appears that most of the degeneracy seen in **Affine** is lifted by the nonnegativity constraints.

**Conic** finds basis vectors whose conic hull best models the input images. The learning algorithm is similar to **Convex** except there is no penalty term on the sum of the activities. The **Conic** representation allows combinations of basis vectors, not just interpolations between them. As a result, the basis vectors found are features rather than templates, as seen in Figure 2. In contrast to **Affine**, the nonnegativity constraint leads to features that are interpretable as correlated strokes. As the number of basis vectors $r$ increases, these features decrease in size until they become individual pixels.

These models were used to classify novel test images. Recognition was accomplished by separately reconstructing the test images with the different digit models and associating the image with the model having the smallest reconstruction error. Figure 3 illustrates an example of classifying a "2" using the conic and convex encodings. The basis vectors are displayed weighted by their activites $v_a$ and the sparsity in the representations can be clearly seen. The bottom part of the figure shows the different reconstructions generated by the various digit models.

With $r = 25$ patterns per digit class, **Convex** incorrectly classified 113 digits out of the 2007 test examples for an overall error rate of 5.6%. This is virtually identical to the performance of $k = 1$ nearest neighbor (112 errors) and linear $r = 25$ PCA models (111 errors). However, scaling up the convex models to $r = 100$ patterns results in an error rate of 4.4% (89 errors). This improvement arises because the larger convex hulls can better represent the overall nonlinear nature of the input

distributions. This is good performance relative to other methods that do not
use prior knowledge of invariances, such as the support vector machine (4.0% [5]).
However, it is not as good as methods that do use prior knowledge, such as nearest
neighbor with tangent distance (2.6% [6]).

On the other hand, **Conic** coding with $r = 25$ results in an error rate of 6.8%
(138 errors). With larger basis sets $r > 50$, **Conic** shows worse performance as the
features shrink to small spots. These results indicate that by itself, **Conic** does not
yield good models; non-trivial correlations still remain in the $v_a$ and also need to
be taken into account. For instance, while the conic basis for "9" can fit some "7"'s
quite well with little reconstruction error, the codes $v_a$ are distinct from when it
fits "9"'s.

## 5  Neural network implementation

**Conic** and **Convex** were described above as matrix factorizations. Alternatively,
the encoding can be performed by a neural network dynamics [7] and the learning
by a synaptic update rule. We describe here the implementation for the **Conic**
network; the **Convex** network is similar. The **Conic** network has a layer of $N$
error neurons $e_i$ and a layer of $r$ encoding neurons $v_a$. The fixed point of the
encoding dynamics

$$\frac{dv_a}{dt} + v_a = \left[\sum_{i=1}^{N} e_i W_{ia} + v_a\right]^+ , \tag{3}$$

$$\frac{de_i}{dt} + e_i = x_i - \sum_{a=1}^{r} W_{ia} v_a , \tag{4}$$

optimizes Eq. (1), finding the best convex encoding of the input $x_i$. The rectification
nonlinearity $[x]^+ = \max(x, 0)$ enforces the nonnegativity constraint. The error
neurons subtract the reconstruction from the input $x_i$. The excitatory connection
from $e_i$ to $v_a$ is equal and opposite to the inhibitory connection from $v_a$ back to $e_i$.
The Hebbian synaptic weight update

$$\Delta W_{ia} = \eta\, e_i v_a \tag{5}$$

is made following convergence of the encoding dynamics for each input, while re-
specting the bound constraints on $W_{ia}$. This performs stochastic gradient descent
on the ensemble reconstruction error with learning rate $\eta$.

## 6  Discussion

**Convex** coding is similar to other locally linear models [8, 9, 10, 11]. Distance to
a convex hull was previously used in nearest neighbor classification [12], though no
learning algorithm was proposed. **Conic** coding is similar to the noisy OR [13, 14]
and harmonium [15] models. The main difference is that these previous models
contain discrete binary variables, whereas **Conic** uses continuous ones. The use of
analog rather than binary variables makes the encoding computationally tractable
and allows for interpolation between basis vectors.

Here we have emphasized the geometrical interpretation of **Convex** and **Conic** coding. They can also be viewed as probabilistic hidden variable models. The inputs $x_i$ are visible while the $v_a$ are hidden variables, and the reconstruction error in Eq. (1) is related to the log likelihood, $\log P(x_i|v_a)$. No explicit model $P(v_a)$ for the hidden variables was used, which limited the quality of the **Conic** models in particular. The feature discovery capabilities of **Conic**, however, make it a promising tool for building hierarchical representations. We are currently working on extending these new coding schemes and learning algorithms to multilayer networks.

We acknowledge the support of Bell Laboratories. We thank C. Burges, C. Cortes, and Y. LeCun for providing us with the USPS database. We are also grateful to K. Clarkson, R. Freund, L. Kaufman, L. Saul, and M. Wright for helpful discussions.

## Footnotes

[1] **Affine** is essentially equivalent to PCA, except that they represent the affine space in different ways. **Affine** represents it with $r$ points chosen from the space. PCA represents the affine space with a single point from the space and $r - 1$ orthonormal directions. This is still a degenerate representation, but PCA fixes it by taking the point to be the sample mean and the $r - 1$ directions to be the eigenvectors of the covariance matrix of $X$ with the largest eigenvalues.

# References

[1] Hinton, GE & Zemel, RS (1994). Autoencoders, minimum description length and Helmholtz free energy. *Advances in Neural Information Processing Systems 6*, 3–10.

[2] Ghahramani, Z (1995). Factorial learning and the EM algorithm. *Advances in Neural Information Processing Systems 7*, 617–624.

[3] Olshausen, BA & Field, DJ (1996). Emergence of simple-cell receptive field properties by learning a sparse code for natural images. *Nature 381*, 607–609.

[4] Le Cun, Y et al. (1989). Backpropagation applied to handwritten zip code recognition. *Neural Comput. 1*, 541–551.

[5] Scholkopf, B, Burges, C, & Vapnik, V (1995). Extracting support data for a given task. *KDD-95 Proceedings*, 252–257.

[6] Simard, P, Le Cun Y & Denker J (1993). Efficient pattern recognition using a new transformation distance. *Advances in Neural Information Processing Systems 5*, 50–58.

[7] Tank, DW & Hopfield, JJ (1986). Simple neural optimization networks: an A/D converter, signal decision circuit, and a linear programming circuit. *IEEE Trans. Circ. Syst.* **CAS-33**, 533–541.

[8] Bezdek, JC, Coray, C, Gunderson, R & Watson J (1981). Detection and characterization of cluster substructure. *SIAM J. Appl. Math.* **40**, 339–357; 358–372.

[9] Bregler, C & Omohundro, SM (1995). Nonlinear image interpolation using manifold learning. *Advances in Neural Information Processing Systems 7*, 973–980.

[10] Hinton, GE, Dayan, P & Revow M (1996). Modeling the manifolds of images of handwritten digits. *IEEE Trans. Neural Networks*, submitted.

[11] Hastie, T, Simard, P & Säckinger E (1995). Learning prototype models for tangent distance. *Advances in Neural Information Processing Systems 7*, 999–1006.

[12] Haas, HPA, Backer, E & Boxma, I (1980). Convex hull nearest neighbor rule. *Fifth Intl. Conf. on Pattern Recognition Proceedings*, 87–90.

[13] Dayan, P & Zemel, RS (1995). Competition and multiple cause models. *Neural Comput. 7*, 565–579.

[14] Saund, E (1995). A multiple cause mixture model for unsupervised learning. *Neural Comput. 7*, 51–71.

[15] Freund, Y & Haussler, D (1992). Unsupervised learning of distributions on binary vectors using two layer networks. *Advances in Neural Information Processing Systems 4*, 912–919.
